# Unsupervised Kernel Dimension Reduction

**Meihong Wang**
Dept. of Computer Science
U. of Southern California
Los Angeles, CA 90089
meihongw@usc.edu

**Fei Sha**
Dept. of Computer Science
U. of Southern California
Los Angeles, CA 90089
feisha@usc.edu

**Michael I. Jordan**
Dept. of Statistics
U. of California
Berkeley, CA
jordan@cs.berkeley.edu

## Abstract

We apply the framework of kernel dimension reduction, originally designed for supervised problems, to unsupervised dimensionality reduction. In this framework, kernel-based measures of independence are used to derive low-dimensional representations that maximally capture information in covariates in order to predict responses. We extend this idea and develop similarly motivated measures for unsupervised problems where covariates and responses are the same. Our empirical studies show that the resulting compact representation yields meaningful and appealing visualization and clustering of data. Furthermore, when used in conjunction with supervised learners for classification, our methods lead to lower classification errors than state-of-the-art methods, especially when embedding data in spaces of very few dimensions.

## 1 Introduction

Dimensionality reduction is an important aspect of many statistical learning tasks. In unsupervised dimensionality reduction, the primary interest is to preserve significant properties of the data in a low-dimensional representation. Well-known examples of this theme include principal component analysis, manifold learning algorithms and their many variants [1–4].

In supervised dimensionality reduction, side information is available to influence the choice of the low-dimensional space. For instance, in regression problems, we are interested in jointly discovering a low-dimensional representation $Z$ of the covariates $X$ *and* predicting well the response variable $Y$ given $Z$. A classical example is Fisher discriminant analysis for binary response variables, which projects $X$ to a one-dimensional line. For more complicated cases, however, one needs to specify a suitable regression function, $\mathbb{E}\left[Y \mid X\right]$, in order to identify $Z$. This is often a challenging task in itself, especially for high-dimensional covariates. Furthermore, one can even argue that this task is cyclically dependent on identifying $Z$, as one of the motivations for identifying $Z$ is that we would hope that the low-dimensional representation can guide us in selecting a good regression function.

To address this dilemma, there has been a growing interest in sufficient dimension reduction (SDR) and related techniques [5–8]. SDR seeks a low-dimensional $Z$ which captures all the dependency between $X$ and $Y$. This is ensured by requiring conditional independence among the three variables; i.e., $X \perp\!\!\!\perp Y \mid Z$. Several classical approaches exist to identify such random vectors $Z$ [6, 9]. Recently, kernel methods have been adapted to this purpose. In particular, *kernel dimensional reduction* (KDR) develops a kernel-based contrast function that measures the degree of conditional independence [7]. Compared to classical techniques, KDR has the significant advantage that it avoids making strong assumptions about the distribution of $X$. Therefore, KDR has been found especially suitable for high-dimensional problems in machine learning and computer vision [8, 10, 11].

In this paper we show how the KDR framework can be used in the setting of unsupervised learning. Our idea is similar in spirit to a classical idea from the neural network literature: we construct

an "autoencoder" or "information bottleneck" where the response variables are the same as the covariates [12, 13]. The key difference is that autoencoders in the neural network literature were based on a specific parametric regression function. By exploiting the SDR and KDR frameworks, on the other hand, we can cast the unsupervised learning problem within a general nonparametric framework involving conditional independence, and in particular as one of optimizing kernel-based measures of independence.

We refer to this approach as "unsupervised kernel dimensionality reduction" (UKDR). As we will show in an empirical investigation, the UKDR approach works well in practice, comparing favorably to other techniques for unsupervised dimension reduction. We assess this via visualization and via building classifiers on the compact representations delivered by these methods. We also provide some interesting analytical links of the UKDR approach to stochastic neighbor embedding (SNE) and $t$-distributed SNE ($t$-SNE) [14, 15].

The paper is organized as follows. In Section 2, we review the SDR framework and discuss how kernels can be used to solve the SDR problem. Additionally, we describe two specific kernel-based measures of independences, elucidating a relationship between these measures. We show how the kernel-based approach can be used for unsupervised dimensionality reduction in Section 3. We report empirical studies in Section 4. Finally, we conclude and comment on possible future directions in Section 5.

**Notation** Random variables are denoted with upper-case characters such as $X$ and $Y$. To refer to their specific values, if vectorial, we use bold lower-case such as $\boldsymbol{x}$ and $\boldsymbol{x}_n$. $x_i$ stands for the $i$-th element of $\boldsymbol{x}$. Matrices are in bold upper-case such as $\boldsymbol{M}$.

## 2 Sufficient dimension reduction and measures of independence with kernels

Discovering statistical (in)dependencies among random variables is a classical problem in statistics; examples of standard measures include Spearman's $\rho$, Kendall's $\tau$ and Pearson's $\chi^2$ tests. Recently, there have been a growing interest in computing measures of independence in Reproducing Kernel Hilbert spaces (RKHSs) [7, 16]. Kernel-based (and other nonparametric) methods detect nonlinear dependence in random variables without assuming specific relationships among them. In particular, the resulting independence measures attain minimum values when random variables are independent. These methods were originally developed in the context of independent component analysis [17] and have found applications in a variety of other problems, including clustering, feature selection, and dimensionality reduction [7, 8, 18–21].

We will be applying these approaches to unsupervised dimensionality reduction. Our proposed techniques aim to yield low-dimensional representation which is "maximally" dependent on the original high-dimensional inputs—this will be made precise in a later section. To this end, we first describe briefly kernel-based measures of (conditional) independence, focusing on how they are applied to supervised dimensionality reduction.

### 2.1 Kernel dimension reduction for supervised learning

In supervised dimensionality reduction for classification and regression, the response variable, $Y \in \mathcal{Y}$, provides side information about the covariates, $X \in \mathcal{X}$. In a basic version of this problem we seek a linear projection $\boldsymbol{B} \in \mathbb{R}^{D \times M}$ to project $X$ from D-dimensional space to a $M$-dimensional subspace. We would like the low-dimensional coordinates $Z = \boldsymbol{B}^\top X$ to be as predictive about $Y$ as $X$ is; i.e., $\mathbb{E}[Y \mid \boldsymbol{B}^\top X] = \mathbb{E}[Y \mid X]$. Intuitively, knowing $Z$ is sufficient for the purpose of regressing $Y$.

This problem is referred to as *sufficient dimension reduction* (SDR) in statistics, where it has been the subject of a large literature [22]. In particular, SDR seeks a projection $\boldsymbol{B}$ such that,

$$X \perp\!\!\!\perp Y \mid \boldsymbol{B}^\top X, \quad \text{subject to } \boldsymbol{B}^\top \boldsymbol{B} = \boldsymbol{I}. \tag{1}$$

where $\boldsymbol{I}$ is the $M \times M$ identity matrix. Several methods have been proposed to estimate $\boldsymbol{B}$ [6, 9]. Of special interest is the technique of kernel dimensional reduction (KDR) that is based on assessing conditional independence in RKHS spaces [7]. Concretely, we map the two variables $X$ and $Y$ to the RKHS spaces $\mathcal{F}$ and $\mathcal{G}$ induced by two positive semidefinite kernels $K_X : \mathcal{X} \times \mathcal{X} \to \mathbb{R}$

and $K_Y : \mathcal{Y} \times \mathcal{Y} \to \mathbb{R}$. For any function $g \in \mathcal{G}$, there exists a conditional covariance operator $\mathcal{C}_{YY|X} : \mathcal{G} \to \mathcal{G}$ such that

$$\langle g, \mathcal{C}_{YY|X}\, g \rangle_{\mathcal{G}} = \mathbb{E}\left[\text{var}_{Y|X}[g(Y)|X]\right] \tag{2}$$

calculates the residual errors of predicting $g(Y)$ with $X$ [7, Proposition 3]. Similarly, we can define the conditional covariance operator $\mathcal{C}^{B}_{YY|X}$ for predicting with $\boldsymbol{B}^{\top}X$.

The conditional covariance operator has an important property: for any projection $\boldsymbol{B}$, $\mathcal{C}^{B}_{YY|X} \geq \mathcal{C}_{YY|X}$ where the (partial) order is defined in terms of the trace operator. Moreover, the equality holds if and only if eq. (1) is satisfied. This gives rise to the possibility of using the trace of the operators as a contrast function to estimate $\boldsymbol{B}$.

Concretely, with $N$ samples drawn from $P(X, Y)$, we compute the corresponding kernel matrices $\boldsymbol{K}_{B^{\top}X}$ and $\boldsymbol{K}_Y$. We centralize them with a projection matrix $\boldsymbol{H} = \boldsymbol{I} - 1/N\, \boldsymbol{1}\boldsymbol{1}^{\top}$, where $\boldsymbol{1} \in \mathbb{R}^N$ be the vector whose elements are all ones. The trace of the estimated conditional variance operator $\mathcal{C}^{B}_{YY|X}$ is then defined as follows:

$$\hat{J}_{YY|X}(\boldsymbol{B}^{\top}X, Y) = \text{Trace}\left[\boldsymbol{G}_Y(\boldsymbol{G}_{B^{\top}X} + N\epsilon_N \boldsymbol{I}_N)^{-1}\right], \tag{3}$$

where $\boldsymbol{G}_Y = \boldsymbol{H}\boldsymbol{K}_Y\boldsymbol{H}$ and $\boldsymbol{G}_{B^{\top}X} = \boldsymbol{H}\boldsymbol{K}_{B^{\top}X}\boldsymbol{H}$. $\epsilon_N$ is a regularizer, smoothing the kernel matrix. It should be chosen such that when $N \to +\infty$, $\epsilon_N \to 0$ and $\sqrt{N}\epsilon_N \to +\infty$ to ensure consistency [7]. The minimizer of the conditional independence measure yields the optimal projection $\boldsymbol{B}$ for kernel dimensionality reduction:

$$\boldsymbol{B}_{YY|X} = \arg\min_{\boldsymbol{B}^{\top}\boldsymbol{B}=\boldsymbol{I}} \hat{J}_{YY|X}(\boldsymbol{B}^{\top}X, Y). \tag{4}$$

We defer discussion on choosing kernels as well as numerical optimization to later sections. When it is clear from context, we use $\hat{J}_{YY|X}$ as a shorthand for $\hat{J}_{YY|X}(\boldsymbol{B}^{\top}X, Y)$.

The optimization functional in eq. (3) is not the only way to implement the KDR idea. Indeed, another kernel-based measure of independence that can be optimized in the KDR context is the *Hilbert-Schmidt Independence Criterion* (HSIC) [16]. This is built as the Hilbert-Schmidt norm of the *cross-covariance operator* $\mathcal{C}_{XY}$, defined as $\mathcal{G} \to \mathcal{F}$:

$$\text{cov}(f, g) = \langle f, \mathcal{C}_{XY}g \rangle_{\mathcal{F}} = \mathbb{E}_{XY}\left\{[f(X) - \mathbb{E}_X f(X)]\,[g(Y) - \mathbb{E}_Y g(Y)]\right\}, \tag{5}$$

where the expectations are taken with respect to the joint distribution and the two marginals respectively. It has been shown that for universal kernels such as Gaussian kernels the Hilbert-Schmidt norm of $\mathcal{C}_{XY}$ is zero if and only if $X$ and $Y$ are independent [16]. Given $N$ samples from $P(X, Y)$, the empirical estimate of HSIC is given by (up to a multiplicative constant):

$$\hat{J}_{XY}(X, Y) = \text{Trace}\left[\boldsymbol{H}\boldsymbol{K}_X\boldsymbol{H}\boldsymbol{K}_Y\right], \tag{6}$$

where $\boldsymbol{K}_X$ and $\boldsymbol{K}_Y$ are $\mathbb{R}^{N \times N}$ kernel matrices computed over $X$ and $Y$ respectively. To apply this independence measure to dimensionality reduction, we seek a projection $\boldsymbol{B}$ which maximizes $\hat{J}_{XY}(\boldsymbol{B}^{\top}X, Y)$, such that the low-dimensional coordinates $Z = \boldsymbol{B}^{\top}X$ are maximally correlated with $X$,

$$\boldsymbol{B}_{XY} = \arg\max_{\boldsymbol{B}^{\top}\boldsymbol{B}=\boldsymbol{I}} \hat{J}_{XY}(\boldsymbol{B}^{\top}X, \boldsymbol{Y}) = \arg\max_{\boldsymbol{B}^{\top}\boldsymbol{B}=\boldsymbol{I}} \text{Trace}\left[\boldsymbol{H}\boldsymbol{K}_{B^{\top}X}\boldsymbol{H}\boldsymbol{K}_Y\right]. \tag{7}$$

It is interesting to note that the independence measures in eq. (3) and eq. (6) are similar. In fact, we have been able to find conditions under which they are equivalent, as stated in the following proposition.

**Proposition 1.** *Let $N \to +\infty$ and $\epsilon_N \to 0$. Additionally, assume that the samples are distributed uniformly on the unit sphere. If $\sigma_N \ll \epsilon_N^2$, then up to a constant,*

$$\hat{J}_{YY|X}(\boldsymbol{B}^{\top}\boldsymbol{X}, Y) \approx -c_0 N^2 \epsilon_N^2 \hat{J}_{XY}(\boldsymbol{B}^{\top}\boldsymbol{X}, Y). \tag{8}$$

*Therefore, under these conditions it is equivalent to minimize $\hat{J}_{YY|X}(\boldsymbol{B}^{\top}\boldsymbol{X}, Y)$ or to maximize $\hat{J}_{XY}(\boldsymbol{B}^{\top}\boldsymbol{X}, Y)$. Thus, $\boldsymbol{B}_{XY} \approx \boldsymbol{B}_{YY|X}$.*

**Proof** The proof is sketched in the supplementary material. Note that assuming the norm of $X$ is equal to one is not overly restrictive; in practice, one often needs to normalize data points to control the overall scale.

We note that while the two measures are asymptotically equivalent, they have different computational complexity—computing $\hat{J}_{XY}$ does not involve matrix inversion. Furthermore, $\hat{J}_{XY}$ is slightly easier to use in practice as it does not depend on regularization parameters to smooth the kernel matrices.

The HSIC measure $\hat{J}_{XY}$ is also closely related to the technique of kernel alignment which minimizes the angles between (vectorized) kernel matrices $\boldsymbol{K}_X$ and $\boldsymbol{K}_Y$ [23]. This is equivalent to maximizing $\mathrm{Trace}[\boldsymbol{K}_X \boldsymbol{K}_Y]/(\|\boldsymbol{K}_X\|_F \|\boldsymbol{K}_Y\|_F)$. The alignment technique has been used for clustering data $X$ by assigning cluster labels $Y$ so that the two kernel matrices are maximally aligned. The HSIC measure has also been used for similar tasks [18]. While both $\hat{J}_{YY|X}$ and $\hat{J}_{XY}$ have been used for supervised dimensionality reduction with known values of $Y$, they have not yet been applied to unsupervised dimensionality reduction, which is the direction that we pursue here.

## 3 Unsupervised kernel dimension reduction

In unsupervised dimensionality reduction, the low-dimensional representation $Z$ can be viewed as a compression of $X$. The goal is to identify the $Z$ that captures as much of the information in $X$ as possible. This desideratum has been pursued in the neural network literature where autoencoders learn a pair of encoding and decoding functions, $Z = f(X)$ and $X = g(Z)$. A drawback of this approach is that $f$ and $g$ need to be specified a priori, in terms of number of layers and neurons in neural nets.

Can we leverage the advantages of SDR and KDR to identify $Z$ without specifying $f(X)$ or $g(Z)$? In this section, we describe how this can be done, viewing unsupervised dimensionality reduction as a special type of supervised regression problem. We start by considering the simplest case where $Z$ is a linear projection of $X$. We then consider nonlinear approaches.

### 3.1 Linear unsupervised kernel dimension reduction

Given a random variable $X \in \mathbb{R}^D$, we consider the regression problem $\tilde{X} = f(\boldsymbol{B}^\top X)$ where $\tilde{X}$ is a copy of $X$ and $Z = \boldsymbol{B}^\top X \in \mathbb{R}^M$ is the low-dimensional representation of $X$. Following the framework of SDR and KDR in section 2, we seek $\boldsymbol{B}$ such that $X \perp\!\!\!\perp \tilde{X} \mid \boldsymbol{B}^\top X$. Such $\boldsymbol{B}^\top X$ thus captures all information in $X$ in order to construct itself (i.e., $\tilde{X}$).

With a set of $N$ samples from $P(X)$, the linear projection $\boldsymbol{B}$ can be identified as the minimizer of the following kernel-based measure of independence

$$\min_{\boldsymbol{B}^\top \boldsymbol{B} = \boldsymbol{I}} \quad \hat{J}_{XX|\boldsymbol{B}^\top X} = \mathrm{Trace}\left[\boldsymbol{G}_X(\boldsymbol{G}_{\boldsymbol{B}^\top X} + N\epsilon_N \boldsymbol{I})^{-1}\right], \qquad (9)$$

where $\boldsymbol{G}_X$ and $\boldsymbol{G}_{\boldsymbol{B}^\top X}$ are centralized kernel matrices of $\boldsymbol{K}_X$ and $\boldsymbol{K}_{\boldsymbol{B}^\top X}$ respectively. We can alternatively maximize the corresponding HSIC measure of dependence between $\boldsymbol{B}^\top X$ and $X$

$$\max_{\boldsymbol{B}^\top \boldsymbol{B} = \boldsymbol{I}} \quad \hat{J}_{\boldsymbol{B}^\top X\, X} = \mathrm{Trace}\left[\boldsymbol{G}_X \boldsymbol{G}_{\boldsymbol{B}^\top X}\right]. \qquad (10)$$

We refer collectively to this kernel-based dimension reduction method as linear unsupervised KDR (UKDR) and we use $\hat{J}(\boldsymbol{B}^\top X, X)$ as a shorthand for the independence measure to be either minimized or maximized.

### 3.2 Nonlinear unsupervised kernel dimension reduction

For data with complicated multimodal distributions, linear transformation of the inputs $X$ is unlikely to be sufficiently flexible to reveal useful structures. For example, linear projections can result in overlapping clusters in low-dimensional spaces. For the purpose of better data visualization and exploratory data analysis, we describe several simple yet effective nonlinear extensions to linear UKDR. The main idea is to find a linear subspace embedding of nonlinearly transformed $X$. Let

$h(X) \in \mathbb{R}^H$ denote the nonlinear transformation. The projection $\boldsymbol{B}$ is then computed to optimize $\hat{J}(\boldsymbol{B}^\top h(X), X)$.

**Radial Basis Network (RBN).** In the spirit of neural network autoencoder, one obvious choice of $h(X)$ is to use a network of radial basis functions (RBFs). In this case, $H = N$, the number of samples from $X$. For a sample $\boldsymbol{x}_i$, the $n$-th component of $h(\boldsymbol{x}_i)$ is given by

$$h_n^{\text{RBN}}(\boldsymbol{x}_i) = \exp\{-\|\boldsymbol{x}_i - \boldsymbol{x}_n\|^2/\sigma_n^2\}, \tag{11}$$

where $\boldsymbol{x}_n$ is the center of the $n$-th RBF and $\sigma_n$ is the corresponding bandwidth.

**Random Sparse Feature (RSF).** In this approach we draw $D \times H$ elements of $W$ from a multivariate Gaussian distribution with zero mean and identity covariance matrix. We construct the $k$-th element of $h(X)$ as

$$h_k^{\text{RSF}}(X) = \text{Heaviside}(\boldsymbol{w}_k^\top X - b), \tag{12}$$

where $\boldsymbol{w}_k$ is the $k$-th row of $\boldsymbol{W}$ and $b$ is an adjustable offset term. Heaviside($t$) is the step function that takes the value of 1 when $t > 0$ and the value of 0 otherwise. Note that $b$ controls the sparsity of $h^{\text{RSF}}(X)$, a property that can be computationally advantageous.

Our choice of random matrix $\boldsymbol{W}$ is motivated by earlier work in neural networks with infinite number of hidden units, and recent work in large-scale kernel machines and deep learning kernels [24–26]. In particular, in the limit of $H \to +\infty$, the transformed $X$ induces an RKHS space with the arccos kernel: $h^{\text{RSF}}(\boldsymbol{u})^\top h^{\text{RSF}}(\boldsymbol{v}) = 1 - 1/\pi \cos^{-1}(\boldsymbol{u}^\top \boldsymbol{v}/\|\boldsymbol{u}\|\|\boldsymbol{v}\|)$ [26].

**Nonparametric.** We have also experimented with a setup where $Z$ is not constrained to any parametric form. In particular, we optimize $\hat{J}(Z, X)$ over all possible values $Z \in \mathbb{R}^M$. While more powerful in principle than either linear KDR or the RBF or RSF variants of nonlinear KDR, we have found that empirically that the optimization can get stuck in local optima. However, when initialized with the solutions from the other nonlinear methods, the final solution is generally better.

### 3.3 Choice of kernels

The independence measures $\hat{J}(\boldsymbol{B}^\top X, X)$ are defined via kernels over $\boldsymbol{B}^\top X$ and $X$. A natural choice is a universal kernel, in particular the Gaussian kernel: $K_{\boldsymbol{B}^\top X}(\boldsymbol{x}_i, \boldsymbol{x}_j) = \exp\{-\|\boldsymbol{B}^\top \boldsymbol{x}_i - \boldsymbol{B}^\top \boldsymbol{x}_j\|^2/\sigma_B^2\}$, and similarly for $X$ with a different bandwidth $\sigma_X$. We have also experimented with other types of kernels; in particular we have found the following kernels to be of particular interest.

**Random walk kernel over $X$.** Given $N$ observations, $\{\boldsymbol{x}_1, \boldsymbol{x}_2, \ldots, \boldsymbol{x}_N\}$, we note that the RBN transformed $\boldsymbol{x}_i$ in eq. (11), when properly normalized, can be seen as the probability of random walk from $\boldsymbol{x}_i$ to $\boldsymbol{x}_j$,

$$p_{ij} = P(\boldsymbol{x}_i \to \boldsymbol{x}_j) = \exp\{-\|\boldsymbol{x}_i - \boldsymbol{x}_j\|^2/\sigma_i^2\} / \sum_{j \neq i} \exp\{-\|\boldsymbol{x}_i - \boldsymbol{x}_j\|^2/\sigma_i^2\}. \tag{13}$$

The matrix $\boldsymbol{P}$ with elements of $p_{ij}$ is clearly not symmetric and not positive semidefinite. Nevertheless, a simple transformation $\boldsymbol{K}_X = \boldsymbol{P}\boldsymbol{P}^\top$ turns it into a positive semidefinite kernel. Intuitively, the values of $p_{ij}$ describe local structures around $\boldsymbol{x}_i$ [14]. Thus $K_X(\boldsymbol{x}_i, \boldsymbol{x}_j) = \sum_k p_{ik} p_{jk}$ measures the similarity between $\boldsymbol{x}_i$ and $\boldsymbol{x}_j$ in terms of these local structures.

**Cauchy kernel for $\boldsymbol{B}^\top X$.** A Cauchy kernel is a positive semidefinite kernel and is given by

$$C(\boldsymbol{u}, \boldsymbol{v}) = 1/\left(1 + \|\boldsymbol{u} - \boldsymbol{v}\|^2\right) = \exp\left\{-\log(1 + \|\boldsymbol{u} - \boldsymbol{v}\|^2)\right\}. \tag{14}$$

We define $K_{\boldsymbol{B}^\top X}(\boldsymbol{x}_i, \boldsymbol{x}_j) = C(\boldsymbol{B}^\top \boldsymbol{x}_i, \boldsymbol{B}^\top \boldsymbol{x}_j)$. Intuitively, the Cauchy kernel can be viewed as a Gaussian kernel in the transformed space $\phi(\boldsymbol{B}^\top X)$ such that $\phi(\boldsymbol{x}_i)^\top \phi(\boldsymbol{x}_j) = C(\boldsymbol{x}_i, \boldsymbol{x}_j)$ [27].

These two types of kernels are closely related to $t$-distributed stochastic neighbor embedding ($t$-SNE), a state-of-the-art technique for dimensionality reduction [15]. We discuss the link in the Supplementary Material.

### 3.4 Numerical optimization

We apply gradient-based techniques (with line search) to optimize either independence measure. The techniques constrain the projection matrix $\boldsymbol{B}$ to lie on the Grassman-Stiefel manifold

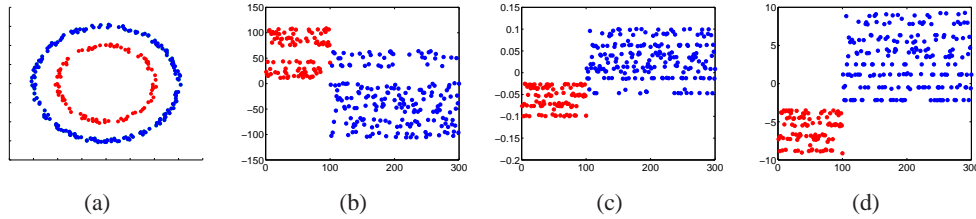

Figure 1: Experiments with synthetic 2D data. (a). Original. (b) 1D embedding by $t$-SNE. (c) and (d) are 1D embeddings by UKDR. They differ in terms of how the embeddings are constrained (see text for details). Vertical axes are the coordinates of 1D embeddings. $t$-SNE failed to separate data. UKDR makes fewer mistakes in (c) and no mistakes in (d).

$B^\top B = I$ [28]. While the optimization is nonconvex, our optimization algorithm works quite well in practice.

The complexity of computing gradients is quadratic in the number of data points as the kernel matrix needs to be computed. Standard tricks—such as chunking—for handling large kernel matrices apply, though our empirical work has not used them. In order to optimize on the Stiefel manifold, computing the search direction from the gradient needs a QR decomposition which depends cubicly on $D$, the original dimensionality. More efficient implementation can bring the complexity to quadratic on $D$ and linearly on $M$, the dimensionality of the low-dimensional space. One simple strategy is to use PCA as a preprocessing step to obtain a moderate $D$.

## 4   Experiments

We compare the performance of our proposed methods for unsupervised kernel dimension reduction (UKDR) to a state-of-the-art method, specifically $t$-distributed stochastic neighbor embedding ($t$-SNE) [15]. $t$-SNE has been shown to excel in many tasks of data visualization and clustering. In addition to visual examination of 2D embedding quality, we also investigate the performance of the resulting low-dimensional representations in classification. In all of the experiments reported in this section, we have used the independence measure $\hat{J}_{B^\top X\,X}(B^\top X, X)$ of eq. (10).

### 4.1   Synthetic example

Our synthetic example contains 300 data points randomly distributed on two rings, shown in Fig. 1(a). We use $t$-SNE and our proposed method to yield 1D embeddings of these data points, plotted in Fig. 1(b)–1(d). The horizontal axis indexes the data points where the first 100 indices correspond to the inner ring.

Fig. 1(b) plots a typical embedding by $t$-SNE where we see that there is significant overlap between the clusters. On the other hand, UKDR is able to generate less overlapped or non-overlapped clusters. In Fig. 1(c), the embedding is computed as the linear projection of the RBN-transformed original data. In Fig. 1(d), the embedding is unconstrained and free to take any value on 1D axis, corresponding to the "nonparametric embedding" presented in section 3.

### 4.2   Images of handwritten digits

Our second data set is a set of 2007 images of USPS handwritten digits [20]. Each image has 256 pixels and is thus represented as a point in $\mathbb{R}^{256}$. We refer to this data set as "USPS-2007." We also sampled a subset of 500 images, 100 each from the digits 1, 2, 3, 4 and 5. Note that images of digit 3 and 5 are often indistinguishable from each other. We refer to this dataset as "USPS-500."

**USPS-500.** Fig. 2 displays a 2D embedding of the 500 images. The colors encode digit categories (which are used only for visualization). The first row was generated with kernel PCA, Laplacian eigenmaps and $t$-SNE. $t$-SNE clearly outperforms the other two in yielding well-separated clusters.

The second row was generated with our UKDR method with Gaussian kernels for both the low-dimensional coordinates $Z$ and $X$. The difference between the three embeddings is whether $Z$ is constrained as a linear projection of the original $X$ (linear UKDR), an RBN-transformed $X$ (RBN UKDR), or a Random Sparse Feature transform of $X$ (RSF UKDR). The Gaussian kernel bandwidths over $Z$ were 0.1, 0.02 and 0.5, respectively. For the RBN transformation of $X$, we selected the bandwidth of each RBF function in eq. (11) with the "perplexity trick" used in SNE and $t$-SNE [15]. The bandwidth for the Gaussian kernel over $X$ was 0.5 for all three plots. While linear UKDR yields reasonably good clusters of the data, RBN UKDR and RSF UKDR yield significantly improved clusterings. Indeed, the quality of the embeddings is on par with that of $t$-SNE.

In the third row of the figure, the embedding $Z$ is constrained to be RSF UKDR. However, instead of using Gaussian kernels (as in the second row), we have used Cauchy kernels. The kernels over $X$ are Gaussian, Random Walk, and Diffusion Map kernels [29], respectively. In general, contrasting to embeddings in the second row, using a Cauchy kernel for the embedding space $Z$ leads to tighter clusters. Additionally, the embeddings by the diffusion map kernel is the most visually appealing one, outperforming $t$-SNE by significantly increasing the gap of digit 1 and 4 from the others.

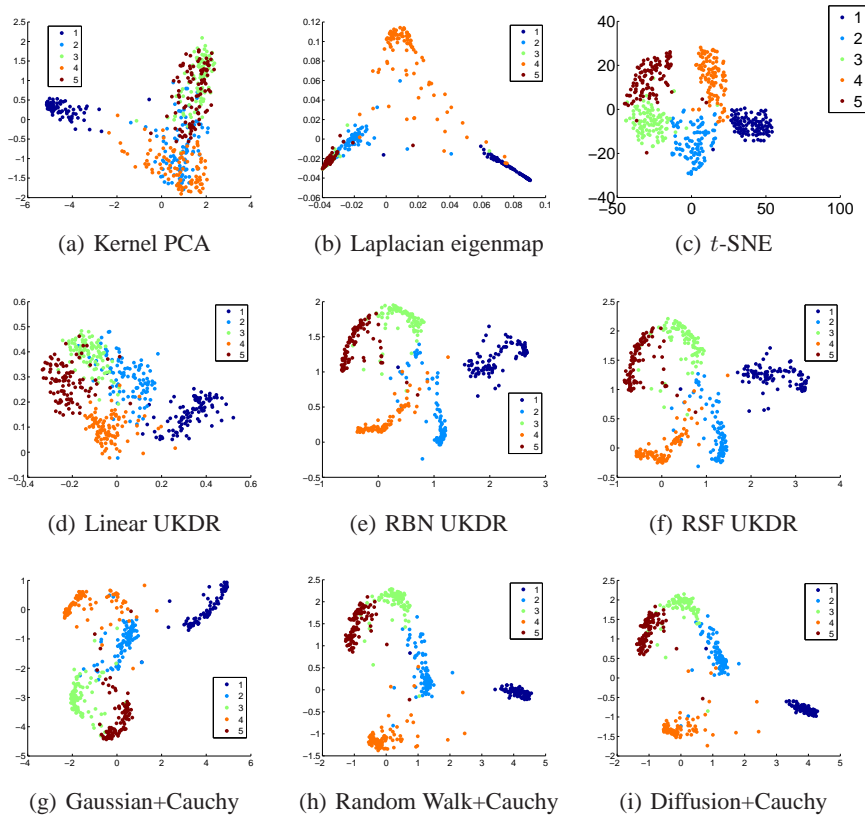

Figure 2: 2D embedding results for the USPS-500 dataset by existing approaches, shown in the first row. Embeddings by UKDR are shown in the bottom two panels.

**Effect of sparsity.** For RSF features computed with eq. (12), the offset constant $b$ can be used to obtain control over the sparsity of the feature vectors. We investigated the effect of the sparsity level on embeddings. We found that a sparsity level as high as 82% still generates reasonable embeddings. Details are reported in the Supplementary Material. Thus RSF features are viable options for handling high-dimensional data for nonlinear UKDR.

**USPS-2007: visualization and classification.** In Fig. 3, we compare the embeddings of $t$-SNE and unsupervised KDR on the full USPS 2007 data set. The data set has many easily confusable pairs of images. Both $t$-SNE and unsupervised KDR lead to visually appealing clustering of data. In the UKDR framework, using an RBN transformation to parameterize the embedding performs slightly better than using the RSF transformation.

| M | 2 | 3 | 5 | 10 | 20 | 50 |
|---|---|---|---|---|---|---|
| UKDR | 11.1 | 11.6 | 9.6 | 9.5 | 8.8 | 7.8 |
| $t$-SNE | 19.8 | 16.8 | 19.3 | 8.4 | 8.2 | 8.1 |
| PCA | 49.3 | 42.2 | 21.5 | 10.03 | 6.7 | 6.6 |

Table 1: Classification errors on the USPS-2007 data set with different dimensionality reduction techniques.

Finally, as another way to assess the quality of the low-dimensional embeddings discovered by these methods, we used these embeddings as inputs to supervised classifiers. The classifier we used was the large-margin nearest-neighbor classifier of [30]. We split the 2007 images into 70% for training and 30% for testing and reporting classification errors. We repeated the random split 50 times and report averaged errors. The results are displayed in table 1 where PCA acts as a baseline. There are several notable findings. First, with very few dimensions (up to and including 5), our UKDR method outperforms both $t$-SNE and PCA significantly. As the dimensionality goes up, $t$-SNE starts to perform better than our method but only marginally. PCA is expected to perform well with very high dimensionality as it recovers pairwise distances the best. The superior classification performance by our method is highly desirable when the target dimensionality is very much constrained.

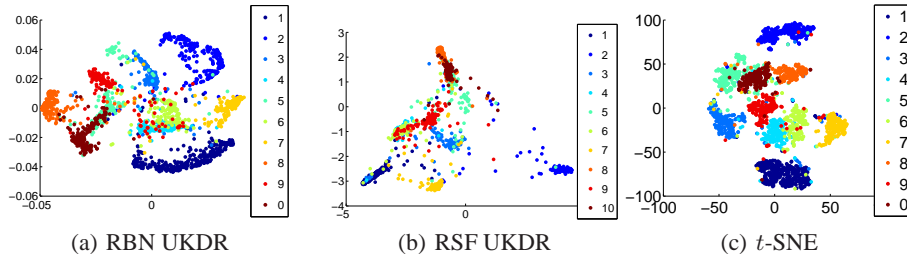

(a) RBN UKDR      (b) RSF UKDR      (c) $t$-SNE

Figure 3: Embeddings of the USPS-2007 data set by our nonlinear UKDR approach and by $t$-SNE. Both methods separate all classes reasonably well. However, using these embeddings as inputs to classifiers suggests that the embedding by nonlinear UKDR is of higher quality.

## 5 Conclusions

We propose a novel technique for unsupervised dimensionality reduction. Our approach is based on kernel dimension reduction. The algorithm identifies low-dimensional representations of input data by optimizing independence measures computed in a reproducing kernel Hilbert space. We study empirically and contrast the performance of our method to that of state-of-the-art approaches. We show that our method yield meaningful and appealing clustering patterns of data. When used for classification, it also leads to significantly lower misclassification.

## Acknowledgements

This work is partially supported by NSF Grant IIS-0957742 and DARPA N10AP20019. F.S. also benefited from discussions with J.P. Zhang, under Fudan University Key Laboratory Senior Visiting Scholar Program.

## References

[1] S. T. Roweis and L. K. Saul. Nonlinear dimensionality reduction by locally linear embedding. *Science*, 290:2323, 2000.

[2] J. B. Tenenbaum, V. Silva, and J. C. Langford. A global geometric framework for nonlinear dimensionality reduction. *Science*, 290:2319, 2000.

[3] C. M. Bishop, M. Svensén, and C. K. I. Williams. GTM: the generative topographic mapping. *Neural Computation*, 10:215–234, 1998.

[4] N. D. Lawrence. Gaussian process latent variable models for visualisation of high dimensional data. In *Advances in Neural Information Processing Systems 16*, pages 329–336. MIT Press, 2004.

[5] R. D. Cook and X. Yin. Dimension reduction and visualization in discriminant analysis (with discussion). *Australian & New Zealand Journal of Statistics*, 43:147–199, 2001.

[6] K. C. Li. Sliced inverse regression for dimension reduction. *Journal of the American Statistical Association*, 86:316–327, 1991.

[7] K. Fukumizu, F. R. Bach, and M. I. Jordan. Kernel dimension reduction in regression. *The Annals of Statistics*, 37:1871–1905, 2009.

[8] J. Nilsson, F. Sha, and M. I. Jordan. Regression on manifolds using kernel dimension reduction. In *Proceedings of the 24th International Conference on Machine Learning*, pages 697–704. ACM, 2007.

[9] K.-C. Li. On principal Hessian directions for data visualization and dimension reduction: another application of Stein's lemma. *Journal of the American Statistical Association*, 86:316–342, 1992.

[10] A. Shyr, R. Urtasun, and M. I. Jordan. Sufficient dimensionality reduction for visual sequence classification. In *Proceedings of Twenty-third IEEE Conference on Computer Vision and Pattern Recognition*, pages 3610–3617, 2010.

[11] Q. Wu, S. Mukherjee, and F. Liang. Localized sliced inverse regression. In *Advances in Neural Information Processing Systems 21*, pages 1785–1792. MIT Press, 2009.

[12] C. M. Bishop et al. *Pattern recognition and machine learning*. Springer New York, 2006.

[13] N. Tishby, F. C. Pereira, and W. Bialek. The information bottleneck method. In *Proceedings of the 37th Annual Allerton Conference on Communication, Control, and Computing*, pages 368–377, 1999.

[14] G. Hinton and S. Roweis. Stochastic neighbor embedding. *Advances in Neural Information Processing Systems 15*, pages 857–864, 2003.

[15] L. van der Maaten and G. Hinton. Visualizing data using t-SNE. *The Journal of Machine Learning Research*, 9:2579–2605, 2008.

[16] A. Gretton, R. Herbrich, A. Smola, O. Bousquet, and B. Schölkopf. Kernel methods for measuring independence. *The Journal of Machine Learning Research*, 6:2075–2129, 2005.

[17] F. R. Bach and M. I. Jordan. Kernel independent component analysis. *The Journal of Machine Learning Research*, 3:1–48, 2003.

[18] L. Song, A. Smola, A. Gretton, and K. M. Borgwardt. A dependence maximization view of clustering. In *Proceedings of the 24th International Conference on Machine Learning*, pages 815–822. ACM, 2007.

[19] L. Song, A. Smola, A. Gretton, K. M. Borgwardt, and J. Bedo. Supervised feature selection via dependence estimation. In *Proceedings of the 24th International Conference on Machine Learning*, pages 823–830. ACM, 2007.

[20] L. Song, A. Smola, K. Borgwardt, and A. Gretton. Colored maximum variance unfolding. *Advances in Neural Information Processing Systems 20*, pages 1385–1392, 2008.

[21] K. Fukumizu, F. R. Bach, and M. I. Jordan. Dimensionality reduction for supervised learning with reproducing kernel Hilbert spaces. *The Journal of Machine Learning Research*, 5:73–99, 2004.

[22] K. P. Adragni and R. D. Cook. Sufficient dimension reduction and prediction in regression. *Philosophical Transactions A*, 367:4385–4405, 2009.

[23] N., J. Kandola, A. Elisseeff, and J. Shawe-Taylor. On kernel-target alignment. In *Advances in Neural Information Processing Systems 14*, pages 367–373. MIT Press, 2002.

[24] C. K. I. Williams. Computation with infinite neural networks. *Neural Computation*, 10:1203–1216, 1998.

[25] A. Rahimi and B. Recht. Random features for large-scale kernel machines. In *Advances in Neural Information Processing Systems 20*, pages 1177–1184. MIT Press, 2008.

[26] Y. Cho and L. Saul. Kernel methods for deep learning. In *Advances in Neural Information Processing Systems 22*, pages 342–350. MIT Press, 2009.

[27] C. Berg, J. P. R. Christensen, and P. Ressel. *Harmonic Analysis on Semigroups*. Springer Verlag, 1984.

[28] A. Edelman, T. A. Arias, and S. T. Smith. The geometry of algorithms with orthogonality constraints. *SIAM J. Matrix Anal. Appl*, 20:303–353, 1998.

[29] B. Nadler, S. Lafon, R. Coifman, and I. G. Kevrekidis. Diffusion maps, spectral clustering and eigenfunctions of Fokker-Planck operators. In *Advances in Neural Information Processing Systems 18*, pages 955–962. MIT Press, 2005.

[30] K. Q. Weinberger and L. K. Saul. Distance metric learning for large margin nearest neighbor classification. *The Journal of Machine Learning Research*, 10:207–244, 2009.

